# A Connectionist Model of the Owl's Sound Localization System

**Daniel J. Rosen***
Department of Psychology
Stanford University
Stanford, CA 94305

**David E. Rumelhart**
Department of Psychology
Stanford University
Stanford, CA 94305

**Eric. I. Knudsen**
Department of Neurobiology
Stanford University
Stanford, CA 94305

## Abstract

We do not have a good understanding of how theoretical principles of learning are realized in neural systems. To address this problem we built a computational model of development in the owl's sound localization system. The structure of the model is drawn from known experimental data while the learning principles come from recent work in the field of brain style computation. The model accounts for numerous properties of the owl's sound localization system, makes specific and testable predictions for future experiments, and provides a theory of the developmental process.

## 1  INTRODUCTION

The barn owl, *Tyto Alba*, has a remarkable ability to localize sounds in space. In complete darkness it catches mice with nearly flawless precision. The owl depends upon this skill for survival, for it is a nocturnal hunter who uses audition to guide

its search for prey (Payne, 1970; Knudsen, Blasdel and Konishi, 1979). Central to the owl's localization system are the precise auditory maps of space found in the owl's optic tectum and in the external nucleus of the inferior colliculus (ICx).

The development of these sensory maps poses a difficult problem for the nervous system, for their accuracy depends upon changing relationships between the animal and its environment. The owl encodes information about the location of a sound source by the phase and amplitude differences with which the sound reaches the owl's two ears. Yet these differences change dramatically as the animal matures and its head grows. The genome cannot "know" in advance precisely how the animal's head will develop – many environmental factors affect this process – so it cannot encode the precise development of the auditory system. Rather, the genome must design the auditory system to adapt to its environment, letting it learn the precise interpretation of auditory cues appropriate for its head and ears.

In order to understand the nature of this developmental process, we built a connectionist model of the owl's sound localization system, using both theoretical principles of learning and knowledge of owl neurophysiology and neuroanatomy.

## 2   THE ESSENTIAL SYSTEM TO BE MODELED

The owl calculates the horizontal component of a sound source location by measuring the interaural time difference (ITD) of a sound as it reaches the two ears (Knudsen and Konishi, 1979). It computes the vertical component of the signal by determining the interaural level difference (ILD) of that same sound (Knudsen and Konishi, 1979). The animal processes these signals through numerous sub-cortical nuclei to form ordered auditory maps of space in both the ICx and the optic tectum. Figure 1 shows a diagram of this neural circuit.

Neurons in both the ICx and the optic tectum are spatially tuned to auditory stimuli. Cells in these nuclei respond to sound signals originating from a restricted region of space in relation to the owl (Knudsen, 1984). Neurons in the ICx respond exclusively to auditory signals. Cells in the optic tectum, on the other hand, encode both auditory and visual sensory maps, and drive the motor system to orient to the location of an auditory or visual signal.

Researchers study the owl's development by systematically altering the animal's sensory experience, usually in one of two ways. They may fit the animal with a sound attenuating earplug, altering its auditory experience, or they may fit the owl with displacing prisms, altering its visual experience.

Disturbance of either auditory or visual cues, during a period when the owl is developing to maturity, causes neural and behavioral changes that bring the auditory map of space back into alignment with the visual map, and/or tune the auditory system to be sensitive to the appropriate range of binaural sound signals. The earplug induced changes take place at the level of the VLVp, where ILD is first computed (Mogdans and Knudsen, 1992). The visually induced adjustment of the auditory maps of space seems to take place at the level of the ICx (Brainard and Knudsen, 1993b). The ability of the owl to adjust to altered sensory signals diminishes over time, and is greatly restricted in adulthood (Knudsen and Knudsen, 1990).

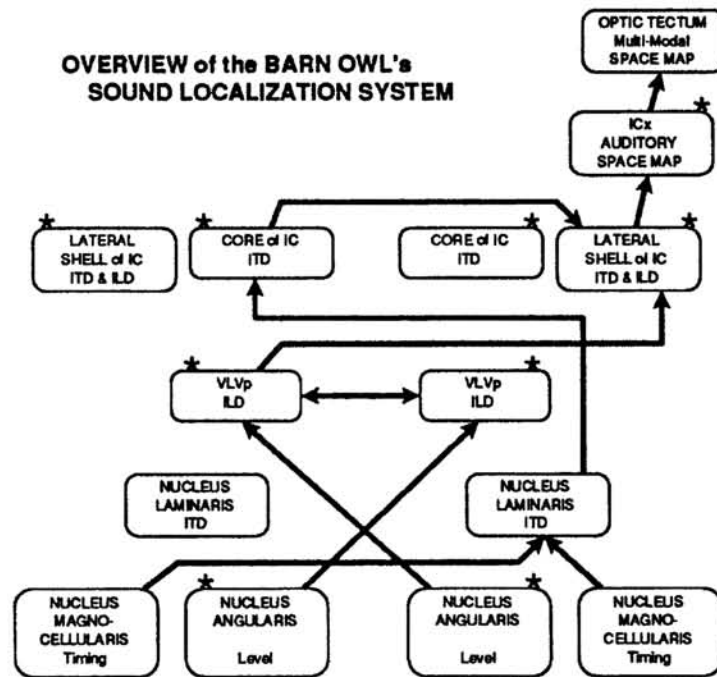

Figure 1: A chart describing the flow of auditory information in the owl's sound localization system. For simplicity, only the connections leading to the one of the bilateral optic tecta are shown. Nuclei labeled with an asterisk (*) are included in the model. Nuclei that process ILD and/or ITD information are so labeled.

## 3    THE NETWORK MODEL

The model has two major components: a network architecture based on the neurobiology of the owl's localization system, as shown in Figure 1, and a learning rule derived from computational learning theory. The elements of the model are standard connectionist units whose output activations are sigmoidal functions of their weighted inputs. The learning rule we use to train the model is not standard. In the following section we describe how and why we derived this rule.

### 3.1    DEFINING THE GOAL OF THE NETWORK

The goal of the network, and presumably the owl, is to accurately map sound signals to sound source locations. The network must discover a model of the world which best captures the relationship between sound signals and sound source locations. Recent work in connectionist learning theory has shown us ways to design networks that search for the model that best fits the data at hand (Buntine and Weigend, 1991; MacKay, 1992; Rumelhart, Durbin, Golden and Chauvin, in press). In this section we apply such an analysis to the localization network.

Table 1: A table showing the mathematical terms used in the analysis.

| TERM | MEANING |
|---|---|
| $\mathcal{M}$ | The Model |
| $\mathcal{D}$ | The Data |
| $P(\mathcal{M}|\mathcal{D})$ | Probability of the Model given the Data |
| $< \vec{x}, \vec{y} >_i$ | The set of $i$ input/target training pairs |
| $\vec{x_i}$ | The input vector for training trial $i$ |
| $\vec{y_i}$ | The target vector for training trial $i$ |
| $\hat{\vec{y}}_i$ | The output vector for training trial $i$ |
| $\hat{y}_{ij}$ | The value of output unit $j$ on training trial $i$ |
| $w_{ij}$ | The weight from unit $j$ to unit $i$ |
| $\eta_j$ | The netinput to unit $j$ |
| $\mathcal{F}(\eta_j)$ | The activation function of unit $j$ evaluated at its netinput |
| $\mathcal{C}$ | The term to be maximized by the network |

## 3.2  DERIVING THE FUNCTION TO BE MAXIMIZED

The network should maximize the probability of the model given the data. Using Bayes' rule we write this probability as:

$$P(\mathcal{M}|\mathcal{D}) = \frac{P(\mathcal{D}|\mathcal{M})P(\mathcal{M})}{P(\mathcal{D})}.$$

Here M represents the model (the units, weights and associated biases) and D represents the data. We define the data as a set of ordered pairs, $[< sound - signal, location - signal >_i]$, which represent the cues and targets normally used to train a connectionist network. In the owl's case the cues are the auditory signals, and the target information is provided by the visual system. (Table 1 lists the mathematical terms we use in this section.)

We simplify this equation by taking the natural logarithm of each side giving:

$$\ln P(\mathcal{M}|\mathcal{D}) = \ln P(\mathcal{D}|\mathcal{M}) + \ln P(\mathcal{M}) - \ln P(\mathcal{D}).$$

Since the natural logarithm is a monotonic transformation, if the network maximizes the second equation it will also maximize the first.

The final term in the equation, $\ln P(\mathcal{D})$, represents the probability of the ordered pairs the network observes. Regardless of which model the network settles upon, this term remains the same – the data are a constant during training. Therefore we can ignore it when choosing a model.

The second term in the equation, $\ln P(\mathcal{M})$, represents the probability of the model. This is the *prior* term in Bayesian analysis and is our estimation of how likely it is that a particular model is true, regardless of the data. We will discuss it below. For now we will concentrate on maximizing $\ln P(\mathcal{D}|\mathcal{M})$.

## 3.3   ASSUMPTIONS ABOUT THE NETWORK'S ENVIRONMENT

We assume that the training data – pairs of stylized auditory and visual signals – are independent of one another and re-write the previous term as:

$$\ln P(\mathcal{D}|\mathcal{M}) \;=\; \sum_i \ln P(<\vec{x},\vec{y}>_i |\mathcal{M}),$$

The $i$ subscript denotes the particular data, or training, pair. We further expand this term to:

$$\ln P(\mathcal{D}|\mathcal{M}) = \sum_i \ln P(\vec{y}_i|\vec{x}_i \wedge \mathcal{M}) + \sum_i \ln P(\vec{x}_i).$$

We ignore the last term, since the sound signals are not dependent on the model. We are left, then, with the task of maximizing $\sum_i \ln P(\vec{y}_i|\vec{x}_i \wedge \mathcal{M})$. It is important to note that $\vec{y}_i$ represents a visual signal, not a localization decision. The network attempts to predict its visual experience given its auditory experience. It does not predict the probability of making an accurate localization decision. If we assume that visual signals provide the target values for the network, then this analysis shows that the auditory map will always follow the visual map, regardless of whether this leads to accurate localization behavior or not. Our assumption is supported by experiments showing that, in the owl, vision *does* guide the formation of auditory spatial maps (Knudsen and Knudsen, 1985; Knudsen, 1988).

Next, we must clarify the relationship between the inputs, $\vec{x}_i$ and the targets, $\vec{y}_i$. We know that the real world is probabilistic – that for a given input there exists some distribution of possible target values. We need to estimate the shape of this distribution. In this case we assume that the target values are binomially distributed – that given a particular sound signal, the visual system did or did not detect a sound source at each point in owl-centered space.

Having made this assumption, we can clarify our interpretation of the network output array, $\vec{\hat{y}}_i$. Each element, $\hat{y}_{ij}$, of this vector represents the activity of output unit $j$ on training trial $i$. We assume that the output activation of each of these units represents the expected value of its corresponding target, $y_{ij}$. In this case the expected value is the mean of a binomial distribution. So the value of each output unit $\hat{y}_{ij}$ represents the probability that a sound signal originated from that particular location. We now write the probability of the data given the model as:

$$P(\vec{y}_i|\vec{x}_i \wedge \mathcal{M}) = \prod_j \hat{y}_{ij}^{y_{ij}}(1 - \hat{y}_{ij})^{1-y_{ij}}.$$

Taking the natural log of the probability and summing over all data pairs we get:

$$\mathcal{C} = \sum_i \sum_j y_{ij} \ln \hat{y}_{ij} + (1 - y_{ij})\ln(1 - \hat{y}_{ij})$$

where $\mathcal{C}$ is the term we want to maximize. This is the standard cross-entropy term.

## 3.4   DERIVING THE LEARNING RULE

Having defined our goal we derive a learning rule appropriate to achieving that goal. To determine this rule we compute $\frac{\partial \mathcal{C}}{\partial \eta_j}$ where $\eta_j$ is the net input to a unit. (In these

equations we have dropped the $i$ subscript, which denotes the particular training trial, since this analysis is identical for all trials.) We write this as:

$$\frac{\partial C}{\partial \eta_j} = \frac{(y_j - \hat{y}_j)}{\hat{y}_j(1 - \hat{y}_j)} \frac{\partial \mathcal{F}(\eta_j)}{\partial \eta_j}.$$

where $\partial \mathcal{F}(\eta_j)$ is the derivative of a unit's activation function evaluated at its net input.

Next we choose an appropriate activation function for the output units. The logistic function, $\mathcal{F}(\eta_j) = \frac{1}{(1+e^{-\eta_j})}$, is a good choice for two reasons. First, it is bounded by zero and one. This makes sense since we assume that the probability that a sound signal originated at any one point in space is bounded by zero and one. Second, when we compute the derivative of the logistic function we get the following result:

$$\partial \mathcal{F}(\eta_j) = \mathcal{F}(\eta_j)(1 - \mathcal{F}(\eta_j)) = \hat{y}_j(1 - \hat{y}_j).$$

This term is the variance of a binomial distribution and when we return to the derivative of our cost function, we see that this variance term is canceled by the denominator. The final derivative we use to compute the weight changes at the output units is therefore:

$$\frac{\partial C}{\partial \eta_j} \propto (y_j - \hat{y}_j).$$

The weights to other units in the network are updated according to the standard backpropagation learning algorithm.

## 3.5  SPECIFYING MODEL PRIORS

There are two types of priors in this model. First is the architectural one. We design a fixed network architecture, described in the previous section, based upon our knowledge of the nuclei involved in the owl's localization system. This is equivalent to setting the prior probability of this architecture to 1, and all others to 0.

We also use a weight elimination prior. This and similar priors may be interpreted as ways to reduce the complexity of a network (Weigend, Huberman and Rumelhart, 1990). The network, therefore, maximizes an expression which is a function of both its error and complexity.

## 3.6  TRAINING

We train the model by presenting it with input to the core of the inferior colliculus (ICc), which encodes interaural phase and time differences (IPD/ITD), and the angular nuclei, which encode sound level. The outputs of the network are then compared to target values, presumed to come from the visual system. The weights are adjusted in order to minimize this difference. We mimic plug training by varying the average difference between the two angular input values. We mimic prism training by systematically changing the target values associated with an input.

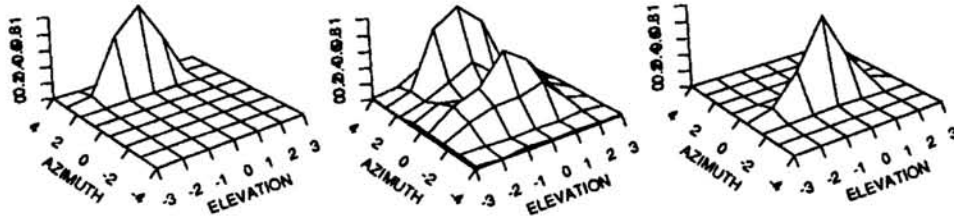

Figure 2: The activity level of ICx units in response to a particular auditory input immediately after simulated prism training was begun (left), midway through training (middle) and after training was completed (right).

## 4   RESULTS and DISCUSSION

The trained network localizes accurately, shows appropriate auditory tuning curves in each of the modeled nuclei, and responds appropriately to manipulations that mimic experiments such as blocking inhibition at the level of the ICx. The network also shows appropriate responses to changing average binaural intensity at the level of the VLVp, the lateral shell and the ICx.

Furthermore, the network exhibits many properties found in the developing owl.. The model appropriately adjusts its auditory localization behavior in simulated earplug experiments and this plasticity takes place at the level of the VLVp. As earplug simulations are begun progressively later in training, the network's ability to adapt to plug training gradually diminishes, following a time course of plasticity qualitatively similar to the sensitive and critical periods described in the owl.

The network adapts appropriately in simulated prism studies and the changes in response to these simulations primarily take place along the lateral shell to ICx connections. As with the plug studies, the network's ability to adapt to prisms diminishes over time. However, unlike the mature owl, a highly trained network retains the ability to adapt in a simulated prism experiment.

We also discovered that the principally derived learning rule better models intermediate stages of prism adjustment than does a standard back-propagation network. Brainard and Knudsen (1993a) report observing two peaks of activity across the tectum in response to an auditory stimulus during prism training – one corresponding to the pre-training response and one corresponding to the newly learned response. Over time the pre-trained response diminishes while the newly learned one grows. As shown in Figure 2, the network exhibits this same pattern of learning. Networks we trained under a standard back-propagation learning algorithm do not. Such a

result lends support to the idea that the owl's localization system is computing a function similar to the one the network was designed to learn.

In addition to accounting for known data, the network predicts results of experiments it was not designed to mimic. Specifically, the network accurately predicted that removal of the animal's facial ruff, which causes ILD to vary with azimuth instead of elevation, would have no effect on the animal's response to varying ILD.

The network accomplishes the goals for which it was designed. It accounts for much, though not all, of the developmental data, it makes testable predictions for future experiments, and since we derived the learning rule in a principled fashion, the network provides us with a specific theory of the owl's sound localization system.

## Footnotes

*Current address: Keck Center for Integrative Neuroscience, UCSF, 513 Parnassus Ave., San Francisco, CA 94143-0444.

## References

Brainard, M. S., & Knudsen, E. I. (1993a). Dynamics of the visual calibration of the map of interaural time difference in the barn owl's optic tectum. *Society for Neuroscience Abstracts, 19*, 369.8.

Brainard, M. S., & Knudsen, E. I. (1993b). Experience-dependent plasticity in the inferior colliculus: a site for visual calibration of the neural representation of auditory space in the barn owl. *The Journal of Neuroscience, 13*, 4589–4608.

Buntine, W. L., & Weigend, A. S. (1991). Bayesian back-propagation. *Complex Systems, 5*, 603–612.

Knudsen, E. (1984). Auditory properties of space-tuned units in owl's optic tectum. *Journal of Neurophysiology, 52*(4), 709–723.

Knudsen, E. (1988). Early blindness results in a degraded auditory map of space in the optic tectum of the barn owl. *Proceedings of the National Academy of Science, U.S.A., 85*, 6211–6214.

Knudsen, E., Blasdel, G., & Konishi, M. (1979). Sound localization by the barn owl (tyto alba) measured with the search coil technique. *The Journal of Comparative Physiology A, 133*, 1–11.

Knudsen, E., & Knudsen, P. (1985). Vision guides the adjustment of auditory localization in young barn owls. *Science, 230*, 545–548.

Knudsen, E., & Knudsen, P. (1990). Sensitive and critical periods for visual calibration of sound localization by barn owls. *The Journal of Neuroscience, 10*(1), 222–232.

MacKay, D. J. (1992). *Bayesian Methods for Adaptive Models.* Unpublished doctoral dissertation, California Institute of Technology, Pasadena, California.

Mogdans, J., & Knudsen, E. I. (1992). Adaptive adjustment of unit tuning to sound localization cues in response to monaural occlusion in developing owl optic tectum. *The Journal of Neuroscience, 12*, 3473–3484.

Payne, R. S. (1970). Acoustic location of prey by barn owls (*tyto alba*). *The Journal of Experimental Biology, 54*, 535–573.

Rumelhart, D. E., Durbin, R., Golden, R., & Chauvin, Y. (in press). Backpropagation: The theory. In Y. Chauvin & D. E. Rumelhart (Eds.), *Backpropagation: Theory, Architectures and Applications.* Hillsdale, N.J.: Lawrence Earlbaum Associates.

Weigend, A. S., Huberman, B. A., & Rumelhart, D. E. (1990). Predicting the future: A connectionist approach. *International Journal of Neural Systems, 1*, 193–209.
